# Pointwise Tracking the Optimal Regression Function

**Ran El-Yaniv** and **Yair Wiener**
Computer Science Department
Technion – Israel Institute of Technology
{rani,wyair}@{cs,tx}.technion.ac.il

## Abstract

This paper examines the possibility of a 'reject option' in the context of least squares regression. It is shown that using rejection it is theoretically possible to learn 'selective' regressors that can $\epsilon$-pointwise track the best regressor in hindsight from the same hypothesis class, while rejecting only a bounded portion of the domain. Moreover, the rejected volume vanishes with the training set size, under certain conditions. We then develop efficient and exact implementation of these selective regressors for the case of linear regression. Empirical evaluation over a suite of real-world datasets corroborates the theoretical analysis and indicates that our selective regressors can provide substantial advantage by reducing estimation error.

## 1 Introduction

Consider a standard least squares regression problem. Given $m$ input-output training pairs, $(x_1, y_1), \ldots, (x_m, y_m)$, we are required to learn a predictor, $\hat{f} \in \mathcal{F}$, capable of generating accurate output predictions, $\hat{f}(x) \in \mathbb{R}$, for any input $x$. Assuming that input-output pairs are i.i.d. realizations of some unknown stochastic source, $P(x, y)$, we would like to choose $\hat{f}$ so as to minimize the standard least squares risk functional,

$$R(\hat{f}) = \int (y - \hat{f}(\mathbf{x}))^2 dP(x, y).$$

Let $f^* = \operatorname{argmin}_{f \in \mathcal{F}} R(f)$ be the optimal predictor in hindsight (based on full knowledge of $P$). A classical result in statistical learning is that under certain structural conditions on $\mathcal{F}$ and possibly on $P$, one can learn a regressor that approaches the average optimal performance, $R(f^*)$, when the sample size, $m$, approaches infinity [1].

In this paper we contemplate the challenge of *pointwise* tracking the optimal predictions of $f^*$ after observing only a finite (and possibly small) set of training samples. It turns out that meeting this difficult task can be made possible by harnessing the 'reject option' compromise from classification. Instead of predicting the output for the entire input domain, the regressor is allowed to abstain from prediction for part of the domain. We present here new techniques for regression with a reject option, capable of achieving *pointwise optimality* on substantial parts of the input domain, under certain conditions.

Section 3 introduces a general strategy for learning selective regressors. This strategy is guaranteed to achieve $\epsilon$-pointwise optimality (defined in Section 2) all through its region of action. This result is proved in Theorem 3.8, which also shows that the guaranteed coverage increases monotonically with the training sample size and converges to 1. This type of guarantee is quite strong, as it ensures tight tracking of individual optimal predictions made by $f^*$, while covering a substantial portion of the input domain.

At the outset, the general strategy we propose appears to be out of reach because accept/reject decisions require the computation of a supremum over a a very large, and possibly infinite hypothesis

subset. In Section 4, however, we show how to compute the strategy for each point of interest using only two constrained ERM calculations. This useful reduction, shown in Lemma 4.2, opens possibilities for efficient implementations of optimal selective regressors whenever the hypothesis class of interest allows for efficient (constrained) ERM (see Definition 4.1).

For the case of linear least squares regression we utilize known techniques for both ERM and constrained ERM and derive in Section 5 exact implementation achieving pointwise optimal selective regression. The resulting algorithm is efficient and can be easily implemented using standard matrix operations including (pseudo) inversion. Theorem 5.3 in this section states a novel pointwise bound on the difference between the prediction of an ERM linear regressor and the prediction of $f^*$ for each individual point. Finally, in Section 6 we present numerical examples over a suite of real-world regression datasets demonstrating the effectiveness of our methods, and indicating that substantial performance improvements can be gained by using selective regression.

***Related work.*** Utilizations of a reject option are quite common in classification where this technique was initiated more than 50 years ago with Chow's pioneering work [2, 3]. However, the reject option is only scarcely and anecdotally mentioned in the context of regression. In [4] a boosting algorithm for regression is proposed and a few reject mechanisms are considered, applied both on the aggregate decision and/or on the underlying weak regressors. A straightforward threshold-based reject mechanism (rejecting low response values) is applied in [5] on top of support vector regression. This mechanism was found to improve false positive rates.

The present paper is inspired and draws upon recent results on selective classification [6, 7, 8], and can be viewed as a natural continuation of the results of [8]. In particular, we adapt the basic definitions of selectivity and the general outline of the derivation and strategy presented in [8].

## 2  Selective regression and other preliminary definitions

We begin with a definition of the following general and standard regression setting. A finite training sample of $m$ labeled examples, $S_m \triangleq \{(x_i, y_i)\}_{i=1}^m \subseteq (\mathcal{X} \times \mathcal{Y})^m$, is observed, where $\mathcal{X}$ is some feature space and $\mathcal{Y} \subseteq \mathbb{R}$. Using $S_m$ we are required to select a regressor $\hat{f} \in \mathcal{F}$, where $\mathcal{F}$ is a fixed *hypothesis class* containing potential regressors of the form $f : \mathcal{X} \to \mathcal{Y}$. It is desired that predictions $\hat{f}(x)$, for unseen instances $x$, will be as accurate as possible. We assume that pairs $(x, y)$, including training instances, are sampled i.i.d. from some *unknown* stochastic source, $P(x, y)$, defined over $\mathcal{X} \times \mathcal{Y}$. Given a loss function, $\ell : \mathcal{Y} \times \mathcal{Y} \to [0, \infty)$, we quantify the prediction quality of any $f$ through its *true error* or *risk*, $R(f)$, defined as its expected loss with respect to $P$,

$$R(f) \triangleq \mathbf{E}_{(x,y)} \left\{ \ell(f(x), y) \right\} = \int \ell(f(x), y) dP(x, y).$$

While $R(f)$ is an unknown quantity, we do observe the *empirical error* of $f$, defined as

$$\hat{R}(f) \triangleq \frac{1}{m} \sum_{i=1}^m \ell(f(x_i), y_i).$$

Let $\hat{f} \triangleq \arg\inf_{f \in \mathcal{F}} \hat{R}(f)$ be the *empirical risk minimizer (ERM)*, and $f^* \triangleq \arg\inf_{f \in \mathcal{F}} R(f)$, the *true risk minimizer*.

Next we define *selective regression* using the following definitions, which are taken, as is, from the selective classification setting of [6]. Here again, we are given a training sample $S_m$ as above, but are now required to output a *selective regressor* defined to be a pair $(f, g)$, with $f \in \mathcal{F}$ being a standard regressor, and $g : \mathcal{X} \to \{0, 1\}$ is a *selection function*, which is served as qualifier for $f$ as follows. For any $x \in \mathcal{X}$,

$$(f, g)(x) \triangleq \begin{cases} reject, & \text{if } g(x) = 0; \\ f(x), & \text{if } g(x) = 1. \end{cases} \tag{1}$$

Thus, the selective regressor abstains from prediction at a point $x$ iff $g(x) = 0$. The general performance of a selective regressor is characterized in terms of two quantities: *coverage* and *risk*. The *coverage* of $(f, g)$ is

$$\Phi(f, g) \triangleq \mathbf{E}_P \left[ g(x) \right].$$

The true risk of $(f, g)$ is the risk of $f$ restricted to its region of activity as qualified by $g$, and normalized by its coverage,

$$R(f, g) \triangleq \frac{\mathbf{E}_P\left[\ell(f(x), y) \cdot g(x)\right]}{\Phi(f, g)}.$$

We say that the selective regressor $(f, g)$ is $\epsilon$-*pointwise optimal* if

$$\forall x \in \{x \in \mathcal{X} : g(x) = 1\}, \quad |f(x) - f^*(x)| \leq \epsilon.$$

Note that pointwise optimality is a considerably stronger property than risk, which only refers to average performance.

We define a (standard) distance metric over the hypothesis class $\mathcal{F}$. For any probability measure $\mu$ on $\mathcal{X}$, let $L_2(\mu)$ be the Hilbert space of functions from $\mathcal{X}$ to $\mathbb{R}$, with the inner product defined as

$$\langle f, g \rangle \triangleq \mathbf{E}_{\mu(x)} f(x) g(x).$$

The distance function induced by the inner product is

$$\rho(f, g) \triangleq \| f - g \| = \sqrt{\langle f - g, f - g \rangle} = \sqrt{\mathbf{E}_{\mu(x)} \left(f(x) - g(x)\right)^2}.$$

Finally, for any $f \in \mathcal{F}$ we define a ball in $\mathcal{F}$ of radius $r$ around $f$,

$$\mathcal{B}(f, r) \triangleq \{f' \in \mathcal{F} : \rho(f, f') \leq r\}.$$

## 3 Pointwise optimality with bounded coverage

In this section we analyze the following strategy for learning a selective regressor, which turns out to ensure $\epsilon$-pointwise optimality with monotonically increasing coverage (with $m$). We call it a strategy rather than an algorithm because it is not at all clear at the outset how to implement it. In subsequent sections we develop efficient and precise implementation for linear regression.

We require the following definition. For any hypothesis class $\mathcal{F}$, target hypothesis $f \in \mathcal{F}$, distribution $P$, sample $S_m$, and real $r > 0$, define,

$$\mathcal{V}(f, r) \triangleq \{f' \in \mathcal{F} : R(f') \leq R(f) + r\} \quad \text{and} \quad \hat{\mathcal{V}}(f, r) \triangleq \left\{f' \in \mathcal{F} : \hat{R}(f') \leq \hat{R}(f) + r\right\}. \tag{2}$$

---

**Strategy 1** A learning strategy for $\epsilon$-pointwise optimal selective regressors

**Input:** $S_m, m, \delta, \mathcal{F}, \epsilon$
**Output:** A selective regressor $(\hat{f}, g)$ achieving $\epsilon$-pointwise optimality
  1: Set $\hat{f} = ERM(\mathcal{F}, S_m)$, i.e., $\hat{f}$ is any empirical risk minimizer from $\mathcal{F}$
  2: Set $G = \hat{\mathcal{V}}\left(\hat{f}, \left(\sigma(m, \delta/4, \mathcal{F})^2 - 1\right) \cdot \hat{R}(\hat{f})\right)$   /* see Definition 3.3 and (2) */
  3: Construct $g$ such that $g(x) = 1 \iff \forall f' \in G \quad |f'(x) - \hat{f}(x)| < \epsilon$

---

For the sake of brevity, throughout this section we often write $f$ instead of $f(x)$, where $f$ is any regressor. The following Lemma 3.1 is based on the proof of Lemma A.12 in [9].

**Lemma 3.1** ([9]). *For any $f \in \mathcal{F}$. Let $\ell : \mathcal{Y} \times \mathcal{Y} \to [0, \infty)$ be the squared loss function and $\mathcal{F}$ be a convex hypothesis class. Then, $\mathbf{E}_{(x,y)}(f^*(x) - y)(f(x) - f^*(x)) \geq 0$.*

**Lemma 3.2.** *Under the same conditions of Lemma 3.1, for any $r > 0$, $\mathcal{V}(f^*, r) \subseteq \mathcal{B}\left(f^*, \sqrt{r}\right)$.*

*Proof.* If $f \in \mathcal{V}(f^*, r)$, then by definition,

$$R(f) \leq R(f^*) + r. \tag{3}$$

$$\begin{aligned}
R(f) - R(f^*) &= \mathbf{E}\left\{\ell(f, y) - \ell(f^*, y)\right\} = \mathbf{E}\left\{(f - y)^2 - (f^* - y)^2\right\} \\
&= \mathbf{E}\left\{(f - f^*)^2 - 2(y - f^*)(f - f^*)\right\} = \rho^2(f, f^*) + 2\mathbf{E}(f^* - y)(f - f^*).
\end{aligned}$$

Applying Lemma 3.1 and (3) we get, $\rho(f, f^*) \leq \sqrt{R(f) - R(f^*)} \leq \sqrt{r}$. $\qquad \square$

**Definition 3.3** (**Multiplicative Risk Bounds**)**.** *Let $\sigma_\delta \triangleq \sigma\,(m, \delta, \mathcal{F})$ be defined such that for any $0 < \delta < 1$, with probability of at least $1 - \delta$ over the choice of $S_m$ from $P^m$, any hypothesis $f \in \mathcal{F}$ satisfies*

$$R(f) \leq \hat{R}(f) \cdot \sigma\,(m, \delta, \mathcal{F})\,.$$

*Similarly, the reverse bound , $\hat{R}(f) \leq R(f) \cdot \sigma\,(m, \mathcal{F}, \delta)$, holds under the same conditions.*

**Remark 3.1.** The purpose of Definition 3.3 is to facilitate the use of any (known) risk bound as a plug-in component in subsequent derivations. We define $\sigma$ as a multiplicative bound, which is common in the treatment of unbounded loss functions such as the squared loss (see discussion by Vapnik in [10], page 993). Instances of such bounds can be extracted, e.g., from [11] (Theorem 1), and from bounds discussed in [10]. We also developed the entire set of results that follow while relying on additive bounds, which are common when using bounded loss functions. These developments will be presented in the full version of the paper.

The proof of the following lemma follows closely the proof of Lemma 5.3 in [8]. However, it considers a multiplicative risk bound rather than additive.

**Lemma 3.4.** *For any $r > 0$, and $0 < \delta < 1$, with probability of at least $1 - \delta$,*

$$\hat{\mathcal{V}}(\hat{f}, r) \subseteq \mathcal{V}\left(f^*, (\sigma_{\delta/2}^2 - 1) \cdot R(f^*) + r \cdot \sigma_{\delta/2}\right).$$

**Lemma 3.5.** *Let $\mathcal{F}$ be a convex hypothesis space, $\ell : \mathcal{Y} \times \mathcal{Y} \to [0, \infty)$, a convex loss function, and $\hat{f}$ be an ERM. Then, with probability of at least $1 - \delta/2$, for any $x \in \mathcal{X}$,*

$$|f^*(x) - \hat{f}(x)| \leq \sup_{f \in \hat{\mathcal{V}}\left(\hat{f}, (\sigma_{\delta/4}^2 - 1) \cdot \hat{R}(\hat{f})\right)} |f(x) - \hat{f}(x)|.$$

*Proof.* Applying the multiplicative risk bound, we get that with probability of at least $1 - \delta/4$,

$$\hat{R}(f^*) \leq R(f^*) \cdot \sigma_{\delta/4}.$$

Since $f^*$ minimizes the true error, $R(f^*) \leq R(\hat{f})$. Applying the multiplicative risk bound on $\hat{f}$, we know also that with probability of at least $1 - \delta/4$, $R(\hat{f}) \leq \hat{R}(\hat{f}) \cdot \sigma_{\delta/4}$. Combining the three inequalities by using the union bound we get that with probability of at least $1 - \delta/2$,

$$\hat{R}(f^*) \leq \hat{R}(\hat{f}) \cdot \sigma_{\delta/4}^2 = \hat{R}(\hat{f}) + \left(\sigma_{\delta/4}^2 - 1\right) \cdot \hat{R}(\hat{f}).$$

Hence, with probability of at least $1 - \delta/2$ we get $f^* \in \hat{\mathcal{V}}\left(\hat{f}, (\sigma_{\delta/4}^2 - 1) \cdot \hat{R}(\hat{f})\right)$ □

Let $G \subseteq \mathcal{F}$. We generalize the concept of *disagreement set* [12, 6] to real-valued functions. The *$\epsilon$-disagreement set* w.r.t. $G$ is defined as

$$DIS_\epsilon(G) \triangleq \{x \in \mathcal{X} : \exists f_1, f_2 \in G \quad s.t. \quad |f_1(x) - f_2(x)| \geq \epsilon\}\,.$$

For any $G \subseteq \mathcal{F}$, distribution $P$, and $\epsilon > 0$, we define $\Delta_\epsilon G \triangleq Pr_P\{DIS_\epsilon(G)\}$. In the following definition we extend Hanneke's disagreement coefficient [13] to the case of real-valued functions.[1]

**Definition 3.6** (**$\epsilon$-disagreement coefficient**)**.** *The $\epsilon$-disagreement coefficient of $\mathcal{F}$ under $P$ is,*

$$\theta_\epsilon \triangleq \sup_{r > r_0} \frac{\Delta_\epsilon \mathcal{B}(f^*, r)}{r}. \tag{4}$$

*Throughout this paper we set $r_0 = 0$. Our analyses for arbitrary $r_0 > 0$ will be presented in the full version of this paper.*

The proof of the following technical statement relies on the same technique used for the proof of Theorem 5.4 in [8].

**Lemma 3.7.** *Let $\mathcal{F}$ be a convex hypothesis class, and assume $\ell : \mathcal{Y} \times \mathcal{Y} \to [0, \infty)$ is the squared loss function. Let $\epsilon > 0$ be given. Assume that $\mathcal{F}$ has $\epsilon$-disagreement coefficient $\theta_\epsilon$. Then, for any $r > 0$ and $0 < \delta < 1$, with probability of at least $1 - \delta$,*

$$\Delta_\epsilon \hat{\mathcal{V}}(\hat{f}, r) \leq \theta_\epsilon \sqrt{\left( \sigma_{\delta/2}^2 - 1 \right) \cdot R(f^*) + r \cdot \sigma_{\delta/2}}.$$

The following theorem is the main result of this section, showing that Strategy 1 achieves $\epsilon$-pointwise optimality with a meaningful coverage that converges to 1. Although $R(f^*)$ in the bound (5) is an unknown quantity, it is still a constant, and as $\sigma$ approaches 1, the coverage lower bound approaches 1 as well. When using a typical additive risk bound, $R(f^*)$ disappears from the RHS.

**Theorem 3.8.** *Assume the conditions of Lemma 3.7 hold. Let $(f, g)$ be the selective regressor chosen by Strategy 1. Then, with probability of at least $1 - \delta$,*

$$\Phi(f, g) \geq 1 - \theta_\epsilon \sqrt{\left( \sigma_{\delta/4}^2 - 1 \right) \cdot \left( R(f^*) + \sigma_{\delta/4} \cdot \hat{R}(\hat{f}) \right)} \qquad (5)$$

*and*

$$\forall x \in \{x \in \mathcal{X} : g(x) = 1\} \quad |f(x) - f^*(x)| < \epsilon.$$

*Proof.* According to Strategy 1, if $g(x) = 1$ then $\sup_{f \in \hat{\mathcal{V}}(\hat{f}, (\sigma_{\delta/4}^2 - 1) \cdot \hat{R}(\hat{f}))} |f(x) - \hat{f}(x)| < \epsilon$. Applying Lemma 3.5 we get that, with probability of at least $1 - \delta/2$,

$$\forall x \in \{x \in \mathcal{X} : g(x) = 1\} \quad |f(x) - f^*(x)| < \epsilon.$$

Since $\hat{f} \in \hat{\mathcal{V}}\left( \hat{f}, (\sigma_{\delta/4}^2 - 1) \cdot \hat{R}(\hat{f}) \right) = G$ wet get

$$
\begin{aligned}
\Phi(f, g) &= \mathbf{E}\{g(X)\} = \mathbf{E}\left\{ \mathbb{I}\left( \sup_{f \in G} |f(x) - \hat{f}(x)| < \epsilon \right) \right\} \\
&= 1 - \mathbf{E}\left\{ \mathbb{I}\left( \sup_{f \in G} |f(x) - \hat{f}(x)| \geq \epsilon \right) \right\} \\
&\geq 1 - \mathbf{E}\left\{ \mathbb{I}\left( \sup_{f_1, f_2 \in G} |f_1(x) - f_2(x)| \geq \epsilon \right) \right\} = 1 - \Delta_\epsilon G.
\end{aligned}
$$

Applying Lemma 3.7 and the union bound we conclude that with probability of at least $1 - \delta$,

$$\Phi(f, g) = \mathbf{E}\{g(X)\} \geq 1 - \theta_\epsilon \sqrt{\left( \sigma_{\delta/4}^2 - 1 \right) \cdot \left( R(f^*) + \sigma_{\delta/4} \cdot \hat{R}(\hat{f}) \right)}.$$

$\square$

# 4 Rejection via constrained ERM

In Strategy 1 we are required to track the supremum of a possibly infinite hypothesis subset, which in general might be intractable. The following Lemma 4.2 reduces the problem of calculating the supremum to a problem of calculating a constrained ERM for two hypotheses.

**Definition 4.1** (**constrained ERM**). *Let $x \in \mathcal{X}$ and $\epsilon \in \mathbb{R}$ be given. Define,*

$$\hat{f}_{\epsilon, x} \triangleq \underset{f \in \mathcal{F}}{\operatorname{argmin}} \left\{ \hat{R}(f) \quad | \quad f(x) = \hat{f}(x) + \epsilon \right\},$$

*where $\hat{f}(x)$ is, as usual, the value of the unconstrained ERM regressor at point $x$.*

**Lemma 4.2.** *Let $\mathcal{F}$ be a convex hypothesis space, and $\ell : \mathcal{Y} \times \mathcal{Y} \to [0, \infty)$, a convex loss function. Let $\epsilon > 0$ be given, and let $(f, g)$ be a selective regressor chosen by Strategy 1 after observing the training sample $S_m$. Let $\hat{f}$ be an ERM. Then,*

$$g(x) = 0 \quad \Leftrightarrow \quad \hat{R}(\hat{f}_{\epsilon, x}) \leq \hat{R}(\hat{f}) \cdot \sigma_{\delta/4}^2 \quad \vee \quad \hat{R}(\hat{f}_{-\epsilon, x}) \leq \hat{R}(\hat{f}) \cdot \sigma_{\delta/4}^2.$$

*Proof.* Let $G \triangleq \hat{\mathcal{V}}\left(\hat{f}, (\sigma_{\delta/4}^2 - 1) \cdot \hat{R}(\hat{f})\right)$, and assume there exists $f \in G$ such that $|f(x) - \hat{f}(x)| \geq \epsilon$. Assume w.l.o.g. (the other case is symmetric) that $f(x) - \hat{f}(x) = a \geq \epsilon$. Since $\mathcal{F}$ is convex,

$$f' = \left(1 - \frac{\epsilon}{a}\right) \cdot \hat{f} + \frac{\epsilon}{a} \cdot f \in \mathcal{F}.$$

We thus have,

$$f'(x) = \left(1 - \frac{\epsilon}{a}\right) \cdot \hat{f}(x) + \frac{\epsilon}{a} \cdot f(x) = \left(1 - \frac{\epsilon}{a}\right) \cdot \hat{f}(x) + \frac{\epsilon}{a} \cdot \left(\hat{f}(x) + a\right) = \hat{f}(x) + \epsilon.$$

Therefore, by the definition of $\hat{f}_{\epsilon,x}$, and using the convexity of $\ell$, together with Jensen's inequality,

$$
\begin{aligned}
\hat{R}(\hat{f}_{\epsilon,x}) &\leq \hat{R}(f') = \frac{1}{m}\sum_{i=1}^m \ell(f'(x_i), y_i) = \frac{1}{m}\sum_{i=1}^m \ell\left(\left(1 - \frac{\epsilon}{a}\right) \cdot \hat{f}(x_i) + \frac{\epsilon}{a} \cdot f(x_i), y_i\right) \\
&\leq \left(1 - \frac{\epsilon}{a}\right) \cdot \frac{1}{m}\sum_{i=1}^m \ell\left(\hat{f}(x_i), y_i\right) + \frac{\epsilon}{a} \cdot \frac{1}{m}\sum_{i=1}^m \ell\left(f(x_i), y_i\right) \\
&= \left(1 - \frac{\epsilon}{a}\right) \cdot \hat{R}(\hat{f}) + \frac{\epsilon}{a} \cdot \hat{R}(f) \leq \left(1 - \frac{\epsilon}{a}\right) \cdot \hat{R}(\hat{f}) + \frac{\epsilon}{a} \cdot \left(\hat{R}(\hat{f}) \cdot \sigma_{\delta/4}^2\right) \\
&= \hat{R}(\hat{f}) + \frac{\epsilon}{a} \cdot \left(\sigma_{\delta/4}^2 - 1\right) \cdot \hat{R}(\hat{f}) \leq \hat{R}(\hat{f}) \cdot \sigma_{\delta/4}^2.
\end{aligned}
$$

As for the other direction, if $\hat{R}(\hat{f}_{\epsilon,x}) \leq \hat{R}(\hat{f}) \cdot \sigma_{\delta/4}^2$. Then $\hat{f}_{\epsilon,x} \in G$ and $\left|\hat{f}_{\epsilon,x}(x) - \hat{f}(x)\right| = \epsilon$. $\square$

So far we have discussed the case where $\epsilon$ is given, and our objective is to find an $\epsilon$-pointwise optimal regressor. Lemma 4.2 provides the means to compute such an optimal regressor assuming that a method to compute a constrained ERM is available (as is the case for squared loss linear regressors ; see next section). However, as was discussed in [6], in many cases our objective is to explore the entire risk-coverage trade-off, in other words, to get a pointwise bound on $|f^*(x) - f(x)|$, i.e., individually for any test point $x$. The following theorem states such a pointwise bound.

**Theorem 4.3.** *Let $\mathcal{F}$ be a convex hypothesis class, $\ell : \mathcal{Y} \times \mathcal{Y} \to [0, \infty)$, a convex loss function, and let $\hat{f}$ be an ERM. Then, with probability of at least $1 - \delta/2$ over the choice of $S_m$ from $P^m$, for any $x \in \mathcal{X}$,*

$$|f^*(x) - \hat{f}(x)| \leq \sup_{\epsilon \in \mathbb{R}} \left\{|\epsilon| : \hat{R}(\hat{f}_{\epsilon,x}) \leq \hat{R}(\hat{f}) \cdot \sigma_{\delta/4}^2\right\}.$$

*Proof.* Define $\tilde{f} \triangleq \underset{f \in \hat{\mathcal{V}}\left(\hat{f}, (\sigma_{\delta/4}^2 - 1) \cdot \hat{R}(\hat{f})\right)}{\operatorname{argmax}} |f(x) - \hat{f}(x)|$. Assume w.l.o.g (the other case is symmetric) that $\tilde{f}(x) = \hat{f}(x) + a$. Following Definition 4.1 we get $\hat{R}(\hat{f}_{a,x}) \leq \hat{R}(\tilde{f}) \leq \hat{R}(\hat{f}) \cdot \sigma_{\delta/4}^2$. Define $\epsilon' = \sup_{\epsilon \in \mathbb{R}} \left\{|\epsilon| : \hat{R}(\hat{f}_{\epsilon,x}) \leq \hat{R}(\hat{f}) \cdot \sigma_{\delta/4}^2\right\}$. We thus have,

$$\sup_{f \in \hat{\mathcal{V}}\left(\hat{f}, (\sigma_{\delta/4}^2 - 1) \cdot \hat{R}(\hat{f})\right)} |f(x) - \hat{f}(x)| = a \leq \epsilon'.$$

An application of Lemma 3.5 completes the proof. $\square$

We conclude this section with a general result on the monotonicity of the empirical risk attained by constrained ERM regressors. This property, which will be utilized in the next section, can be easily proved using a simple application of Jensen's inequality.

**Lemma 4.4 (Monotonicity).** *Let $\mathcal{F}$ be a convex hypothesis space, $\ell : \mathcal{Y} \times \mathcal{Y} \to [0, \infty)$, a convex loss function, and $0 \leq \epsilon_1 < \epsilon_2$, be given. Then, $\hat{R}(f_{\epsilon_1, x_0}) - \hat{R}(\hat{f}) \leq \frac{\epsilon_1}{\epsilon_2}\left(\hat{R}(\hat{f}_{\epsilon_2, x_0}) - \hat{R}(\hat{f})\right)$. The result also holds for the case $0 \geq \epsilon_1 > \epsilon_2$.*

# 5 Selective linear regression

We now restrict attention to linear least squares regression (LLSR), and, relying on Theorem 4.3 and Lemma 4.4, as well as on known closed-form expressions for LLSR, we derive efficient implementation of Strategy 1 and a new pointwise bound. Let $X$ be an $m \times d$ training sample matrix whose $i$th row, $\mathbf{x}_i \in \mathbb{R}^d$, is a feature vector. Let $\mathbf{y} \in \mathbb{R}^m$ be a column vector of training labels.

**Lemma 5.1 (ordinary least-squares estimate [15]).** *The ordinary least square (OLS) solution of the following optimization problem, $\min_\beta \|X\beta - \mathbf{y}\|^2$, is given by $\hat{\beta} \triangleq (X^T X)^+ X^T \mathbf{y}$, where the sign $^+$ represents the pseudoinverse.*

**Lemma 5.2 (constrained least-squares estimate [15], page 166).** *Let $x_0$ be a row vector and $c$ a label. The constrained least-squares (CLS) solution of the following optimization problem*

$$\text{minimize } \|X\beta - \mathbf{y}\|_2 \quad s.t \quad x_0\beta = c,$$

*is given by $\hat{\beta}_C(c) \triangleq \hat{\beta} + (X^T X)^+ x_0^T (x_0(X^T X)^+ x_0^T)^+ \left( c - x_0\hat{\beta} \right)$, where $\hat{\beta}$ is the OLS solution.*

**Theorem 5.3.** *Let $\mathcal{F}$ be the class of linear regressors, and let $\hat{f}$ be an ERM. Then, with probability of at least $1 - \delta$ over choices on $S_m$, for any test point $x_0$ we have,*

$$|f^*(x_0) - \hat{f}(x_0)| \leq \frac{\|X\hat{\beta} - \mathbf{y}\|}{\|XK\|} \sqrt{\sigma_{\delta/4}^2 - 1}, \qquad \text{where } K = (X^T X)^+ x_0^T (x_0(X^T X)^+ x_0^T)^+.$$

*Proof.* According to Lemma 4.4, for squared loss, $\hat{R}(\hat{f}_{\epsilon,x_0})$ is strictly monotonically increasing for $\epsilon > 0$, and decreasing for $\epsilon < 0$. Therefore, the equation, $\hat{R}(\hat{f}_{\epsilon,x_0}) = \hat{R}(\hat{f}) \cdot \sigma_{\delta/4}^2$, where $\epsilon$ is the unknown, has precisely two solutions for any $\sigma > 1$. Denoting these solutions by $\epsilon_1, \epsilon_2$ we get,

$$\sup_{\epsilon \in \mathbb{R}} \left\{ |\epsilon| : \hat{R}(\hat{f}_{\epsilon,x_0}) \leq \hat{R}(\hat{f}) \cdot \sigma_{\delta/4}^2 \right\} = \max(|\epsilon_1|, |\epsilon_2|).$$

Applying Lemma 5.1 and 5.2 and setting $c = X_0\hat{\beta} + \epsilon$, we obtain,

$$\frac{1}{m}\|X\hat{\beta}_C \left( x_0\hat{\beta} + \epsilon \right) - \mathbf{y}\|^2 = \hat{R}(\hat{f}_{\epsilon,x_0}) = \hat{R}(\hat{f}) \cdot \sigma_{\delta/4}^2 = \frac{1}{m}\|X\hat{\beta} - \mathbf{y}\|^2 \cdot \sigma_{\delta/4}^2.$$

Hence, $\|X\hat{\beta} + XK\epsilon - \mathbf{y}\|^2 = \|X\hat{\beta} - \mathbf{y}\|^2 \cdot \sigma_{\delta/4}^2$, so, $2(X\hat{\beta} - \mathbf{y})^T XK\epsilon + \|XK\|^2\epsilon^2 = \|X\hat{\beta} - \mathbf{y}\|^2 \cdot (\sigma_{\delta/4}^2 - 1)$. We note that by applying Lemma 5.1 on $(X\hat{\beta} - \mathbf{y})^T X$, we get,

$$(X\hat{\beta} - \mathbf{y})^T X = \left( X^T \left( X(X^T X)^+ X^T \mathbf{y} - \mathbf{y} \right) \right)^T = (X^T \mathbf{y} - X^T \mathbf{y})^T = 0.$$

Therefore, $\epsilon^2 = \frac{\|X\hat{\beta} - \mathbf{y}\|^2}{\|XK\|^2} \cdot (\sigma_{\delta/4}^2 - 1)$. Application of Theorem 4.3 completes the proof. $\square$

# 6 Numerical examples

Focusing on linear least squares regression, we empirically evaluated the proposed method. Given a labeled dataset we randomly extracted two disjoint subsets: a training set $S_m$, and a test set $S_n$. The selective regressor $(f, g)$ is computed as follows. The regressor $f$ is an ERM over $S_m$, and for any coverage value $\Phi$, the function $g$ selects a subset of $S_n$ of size $n \cdot \Phi$, including all test points with lowest value of the bound in Theorem 5.3.[2]

We compare our method relative to the following simple and natural 1-nearest neighbor (NN) technique for selection. Given the training set $S_m$ and the test set $S_n$, let $NN(x)$ denote the nearest neighbor of $x$ in $S_m$, with corresponding $\rho(x) \triangleq \sqrt{\|NN(x) - x\|^2}$ distance to $x$. These $\rho(x)$ distances, corresponding to all $x \in S_n$, were used as alternative method to reject test points in decreasing order of their $\rho(x)$ values.

We tested the algorithm on 10 of the 14 LIBSVM [16] regression datasets. From this repository we took all sets that are not too small and have reasonable feature dimensionality.[3] Figure 1 depicts

results obtained for five different datasets, each with training sample size $m = 30$, and test set size $n = 200$. The figure includes a matrix of $2 \times 5$ graphs. Each column corresponds to a single dataset. Each of the graphs on the first row shows the average absolute difference between the selective regressor $(f, g)$ and the optimal regressor $f^*$ (taken as an ERM over the entire dataset) as a function of coverage, where the average is taken over the accepted instances. Our method appears in solid red line, and the baseline NN method, in dashed black line. Each curve point is an average over 200 independent trials (error bars represent standard error of the mean). It is evident that for all datasets the average distance monotonically increases with coverage. Furthermore, in all cases the proposed method significantly outperforms the NN baseline.

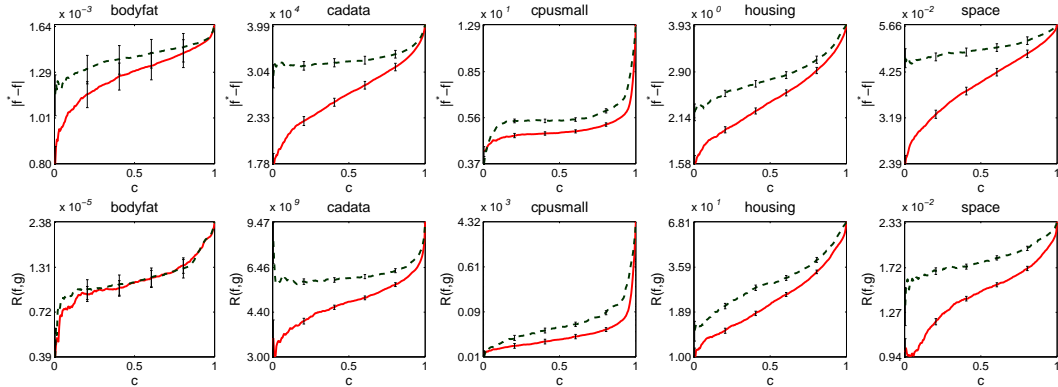

Figure 1: (top row) absolute difference between the selective regressor $(f, g)$ and the optimal regressor $f^*$. (bottom row) test error of selective regressor $(f, g)$. Our proposed method in solid red line and the baseline method in dashed black line. In all curves the $y$-axis has logarithmic scale.

Each of the graphs in the second row shows the test error of the selective regressor $(f, g)$ as a function of coverage. This curve is known as the RC (risk-coverage) trade-off curve [6]. In this case we see again that the test error is monotonically increasing with coverage. In four datasets out of the five we observe a clear domination of the entire RC curve, and in one dataset the performance of our method is statistically indistinguishable from that of the NN baseline method.

## 7 Concluding remarks

Rooted in the centuries-old linear least squares method of Gauss and Legendre, regression estimation remains an indispensable routine in statistical analysis, modeling and prediction. This paper proposes a novel rejection technique allowing for a least squares regressor, learned from a finite and possibly small training sample, to pointwise track, within its selected region of activity, the predictions of the globally optimal regressor in hindsight (from the same class). The resulting algorithm, which is motivated and derived entirely from the theory, is efficient and practical.

Immediate plausible extensions are the handling of other types of regressions including regularized, and kernel regression, as well as extensions to other convex loss functions such as the epsilon-insensitive loss. The presence of the $\epsilon$-disagreement coefficient in our coverage bound suggests a possible relation to active learning, since the standard version of this coefficient has a key role in characterizing the efficiency of active learning in classification [17]. Indeed, a formal reduction of active learning to selective classification was recently found, whereby rejected points are precisely those points to be queried in a stream based active learning setting. Moreover, "fast" coverage bounds in selective classification give rise to fast rates in active learning [7]. Borrowing their intuition to our setting, one could consider devising a querying function for active regression that is based on the pointwise bound of Theorem 5.3.

**Acknowledgments**

The research leading to these results has received funding from both Intel and the European Union's Seventh Framework Programme under grant agreement n° 216886.

## Footnotes

[1] Our attemps to utilize a different known extension of the disagreement coefficient [14] were not successful. Specifically, the coefficient proposed there is unbounded for the squared loss function when $\mathcal{Y}$ is unbounded.

[2]We use here the theorem only for ranking test points, so any constant $> 1$ can be used instead of $\sigma_{\delta/4}^2$.

[3]Two datasets having less than 200 samples, and two that have over 150,000 features were excluded.

## References

[1] V. Vapnik. *Statistical learning theory. 1998*. Wiley, New York, 1998.

[2] C.K. Chow. An optimum character recognition system using decision function. *IEEE Trans. Computer*, 6(4):247–254, 1957.

[3] C.K. Chow. On optimum recognition error and reject trade-off. *IEEE Trans. on Information Theory*, 16:41–36, 1970.

[4] B. Kégl. Robust regression by boosting the median. *Learning Theory and Kernel Machines*, pages 258–272, 2003.

[5] Ö. Ayşegül, G. Mehmet, A. Ethem, and H. Türkan. Machine learning integration for predicting the effect of single amino acid substitutions on protein stability. *BMC Structural Biology*, 9.

[6] R. El-Yaniv and Y. Wiener. On the foundations of noise-free selective classification. *The Journal of Machine Learning Research*, 11:1605–1641, 2010.

[7] R. El-Yaniv and Y. Wiener. Active learning via perfect selective classification. *Journal of Machine Learning Research*, 13:255–279, 2012.

[8] R. El-Yaniv and Y. Wiener. Agnostic selective classification. In *Neural Information Processing Systems (NIPS)*, 2011.

[9] W.S. Lee. *Agnostic Learning and Single Hidden Layer Neural Networks*. PhD thesis, Australian National University, 1996.

[10] V.N. Vapnik. An overview of statistical learning theory. *Neural Networks, IEEE Transactions on*, 10(5):988–999, 1999.

[11] R.M. Kil and I. Koo. Generalization bounds for the regression of real-valued functions. In *Proceedings of the 9th International Conference on Neural Information Processing*, volume 4, pages 1766–1770, 2002.

[12] S. Hanneke. A bound on the label complexity of agnostic active learning. In *ICML*, pages 353–360, 2007.

[13] S. Hanneke. *Theoretical Foundations of Active Learning*. PhD thesis, Carnegie Mellon University, 2009.

[14] A. Beygelzimer, S. Dasgupta, and J. Langford. Importance weighted active learning. In *ICML '09: Proceedings of the 26th Annual International Conference on Machine Learning*, pages 49–56. ACM, 2009.

[15] J.E. Gentle. *Numerical linear algebra for applications in statistics*. Springer Verlag, 1998.

[16] C.C. Chang and C.J. Lin. LIBSVM: A library for support vector machines. *ACM Transactions on Intelligent Systems and Technology*, 2:27:1–27:27, 2011. Software available at "http://www.csie.ntu.edu.tw/ cjlin/libsvm".

[17] S. Hanneke. Rates of convergence in active learning. *The Annals of Statistics*, 39(1):333–361, 2011.

